# Hierarchical Mixture of Classification Experts Uncovers Interactions between Brain Regions

**Bangpeng Yao**[1]     **Dirk B. Walther**[2]     **Diane M. Beck**[2,3*]     **Li Fei-Fei**[1*]

[1]Computer Science Department, Stanford University, Stanford, CA 94305
[2]Beckman Institute, University of Illinois at Urbana-Champaign, Urbana, IL 61801
[3]Psychology Department, University of Illinois at Urbana-Champaign, Champaign, IL 61820
{bangpeng,feifeili}@cs.stanford.edu   {walther,dmbeck}@illinois.edu

## Abstract

The human brain can be described as containing a number of functional regions. These regions, as well as the connections between them, play a key role in information processing in the brain. However, most existing multi-voxel pattern analysis approaches either treat multiple regions as one large uniform region or several independent regions, ignoring the connections between them. In this paper we propose to model such connections in an Hidden Conditional Random Field (HCRF) framework, where the classifier of one region of interest (ROI) makes predictions based on not only its voxels but also the predictions from ROIs that it connects to. Furthermore, we propose a structural learning method in the HCRF framework to automatically uncover the connections between ROIs. We illustrate this approach with fMRI data acquired while human subjects viewed images of different natural scene categories and show that our model can improve the top-level (the classifier combining information from all ROIs) and ROI-level prediction accuracy, as well as uncover some meaningful connections between ROIs.

## 1   Introduction

In recent years, machine learning approaches for analyzing fMRI data have become increasingly popular [15, 24, 18, 16]. In these multi-voxel pattern analysis (MVPA) approaches, patterns of voxels are associated with particular stimuli, leading to verifiable predictions about independent test data. Voxels are extracted from previously known regions of interest (ROIs) [15, 31], selected from the brain by some statistical criterion [24], or defined by a sliding window ("searchlight") positioned at each location in the brain in turn [20]. All of these methods, however, ignore the highly interconnected nature of the brain.

Neuroanatomical evidence from macaque monkeys [10] indicates that brain regions involved in visual processing are indeed highly interconnected. Since research on human subjects is largely limited to non-invasive procedures, considerably less is known about interactions between visual areas in the human brain. Here we demonstrate a method of learning the interactions between regions from fMRI data acquired while human subjects view images of natural scenes.

Determining the category of a natural scene (e.g. classifying a scene as a beach, or a forest) is important for many human activities such as navigation or object perception [30]. Despite the large variety of images within and across categories, humans are very good at categorizing natural scenes [27, 9]. In our recent study of natural scene categorization in humans with functional magnetic resonance imaging (fMRI), we discovered that information about natural scene categories is represented in patterns of activity in the parahippocampal place area (PPA), the retrosplenial cortex (RSC), the lateral occipital complex (LOC), and the primary visual cortex (V1) [31]. We demonstrated that this information can be read out from fMRI activity with a linear support vector machine (SVM) classifier.

---

[*]Diane M. Beck and Li Fei-Fei contributed equally to this work.

Given the highly interconnected nature of the brain, however, it is unlikely that these regions encode natural scene categories independently of each other.

As previous ROI-based MVPA methods studies, in [31] we built predictors for each ROI independently, ignoring their interactions. The method in [31] neither explores connections among the ROIs nor uses the connections to build a classifier on top of all ROIs. In this work, **we propose a method for simultaneously learning the voxel patterns associated with natural scene categories in several ROIs and their interactions in a Hidden Conditional Random Field (HCRF) [28] framework**. In our model, the classifier of each ROI makes predictions based on not only its voxels, but also the prediction results of the ROIs that it connects to. Using the same fMRI data set, we also explore a mutual information based method to discover functional connectivity [5]. Our current model differs from [5], however, by applying a generative model to concurrently estimate the structure of connectivity as well as maximize the end behavioral task (in this case, a scene classification task).

Furthermore, **we propose a structural learning method to automatically uncover the structure of the interactions between ROIs for natural scene categorization**, i.e. to decide which ROIs should be and which ones should not be connected. Unlike existing models for functional connectivity, which mostly rely on the correlation of time courses of voxels [23], our approach makes use of the patterns of activity in ROIs as well as the category labels of the images presented to the subjects. Built in the hierarchical framework of HCRF, our structural learning method utilizes information in the voxel values at the bottom layer of the network as well as categorical labels at the top layer. In our method, the connections between each pair of ROIs are evaluated for their potential to improve prediction accuracy, and only those that show improvement will be added to the final structural map.

In the remaining part of this paper, we first elaborate on our model and structural learning approach in Section 2. We discuss related work on MVPA and connectivity analysis in Section 3. Finally, we present experimental results in Section 4 and conclude the paper in Section 5.

## 2 Modeling Interactions of Brain Regions: a HCRF Representation

The brain is highly interconnected, and the nature of the connections determines to a large extent how information is processed in the brain. We model the connections of brain regions in a Hidden Conditional Random Field (HCRF) framework for the task of natural scene categorization and propose a structural learning method to uncover the pattern of connectivity. In the first part of this section we assume that the structural connections between brain regions are already known. We will discuss in Section 2.2 how these connections are automatically learned.

### 2.1 Integrating Information across Brain Regions

Suppose we are given a set of regions of interest (ROIs) and connections between these regions (see the intermediate layer of Fig.1). Existing ROI-based MVPA approaches build a classifier for each ROI independently [15, 24, 18, 16, 31], neglecting the connections between ROIs. It is our objective here to explore the structure of the connections between ROIs to improve prediction accuracy for decoding viewed scene category from fMRI data.

In order to achieve these goals, we propose a Hidden Conditional Random Field (HCRF) model (Fig.1) to allow each ROI to be influenced by the ROIs that it connects to and build a top-level classifier which makes use of information in all ROIs. In this framework, the classifier for one ROI makes prediction based on the voxels in this region as well as the results of the classifiers of its connected ROIs, thereby improving the accuracy of each ROI. In the absence of evidence about the directionality of connections, we assume them to be symmetric, i.e., to allow the information between two ROIs to go in both directions to the same extent. On the technical side, using an undirected model avoids the difficulties of defining a coherent generative process for graph structures in directed models, thereby giving us more flexibility in representing complex patterns [29].

Our model starts with independently trained classifiers for each ROI as in [31] (the bottom layer of Fig.1). Consider an fMRI data set whose individual brain acquisitions are associated with one of $C$ class labels. For an acquisition sample $i$, the decision values of the $C$ independent classifiers are represented as $\mathcal{X}^i = \{\mathbf{X}_1^i, \cdots, \mathbf{X}_M^i\}$, where $M$ is the number of ROIs. $\mathbf{X}_m^i = \{x_{m,1}^i, \cdots, x_{m,C}^i\}$ are the decision values for the $m$-th ROI, where $x_{m,c}^i$ is the probability that region $m$ assigns sample $i$ to the $c$-th class, irrespective of the information in any other ROI.

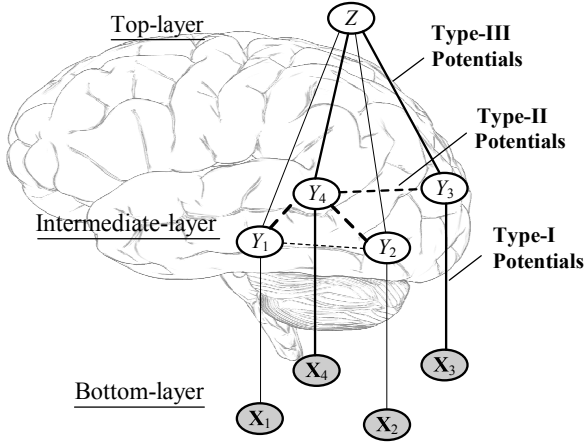

Figure 1: Illustration of the HCRF model for modeling connections between ROIs. Four ROIs, placed figuratively on a schematic brain, are shown here for illustration of the model. Superscripts indexing different samples are omitted in the figure. $Z$ is the category label predicted from all ROIs. $Y_m$, the hidden variable of the model, is the prediction result of the classifier of ROI $m$. $\mathbf{X}_m$ is the output of an independently trained classifier for ROI $m$. Section 2.1 gives details about the three types of connections. In the figure thicker lines represent stronger connections, thinner lines weaker connections. The weights of all connections and connectivity pattern of the type-II potentials are estimated by the model.

Given $\mathbf{X}_m^i$ as input, the classifier for ROI $m$ can directly predict sample $i$ as belonging to the $c^*$-th class if $x_{m,c^*}^i$ is the largest component of $\mathbf{X}_m^i$. However, this method ignores the dependencies between ROIs. To remedy this, our model allows collaborative error-correction over the ROIs by using the given structure of connections (the intermediate layer of Fig.1). Denoting the prediction results of the ROI classifiers as $\mathcal{Y} = \{Y_1, \cdots, Y_M\}$, where $Y_m \in \{1, \cdots, C\}$ is the classifier output for ROI $m$, our model allows for the predictions $Y_m$ and $Y_l$ to interact if ROIs $m$ and $l$ are connected in the given structure (the intermediate layer in Fig.1).

Based on the ROI-level prediction results $\mathcal{Y}$, our model outputs the category label of sample $i$: $Z^i \in \{1, \cdots, C\}$ (the top layer of Fig.1). Furthermore, because we cannot directly observe the prediction of each ROI when acquiring the fMRI data, we treat $\mathcal{Y}$ as hidden variables. The underlying graphical model is shown in Fig.1. To estimate the overall classification probability given the observed voxel values, we marginalize over all possible values of $\mathcal{Y}$. The HCRF model is therefore defined as

$$p(Z^i|\mathcal{X}^i;\boldsymbol{\theta}) = \sum_{\mathcal{Y}} p(Z^i, \mathcal{Y}|\mathcal{X}^i;\boldsymbol{\theta}) = \frac{\sum_{\mathcal{Y}} \exp(\Psi(Z^i, \mathcal{Y}, \mathcal{X}^i;\boldsymbol{\theta}))}{\sum_Z \sum_{\mathcal{Y}} \exp(\Psi(Z, \mathcal{Y}, \mathcal{X}^i;\boldsymbol{\theta}))} \tag{1}$$

where $\boldsymbol{\theta}$ are the parameters of the model, and $\Psi(Z, \mathcal{Y}, \mathcal{X};\boldsymbol{\theta})$ is a potential function parameterized by $\boldsymbol{\theta}$. We define the potential function $\Psi(Z, \mathcal{Y}, \mathcal{X};\boldsymbol{\theta})$ as the weighted sum of edge potential functions defined on every edge $e$ (2-clique) of the model:

$$\Psi(Z, \mathcal{Y}, \mathcal{X};\boldsymbol{\theta}) = \sum_e \theta_e \psi_e(Z, \mathcal{Y}, \mathcal{X}) \tag{2}$$

As shown in Fig.1, there are three types of potentials which describe different edges in the model:

**Type-I Potential** $e = (\mathbf{X}_m, Y_m)$. Such edges model the distribution of class labels of different ROIs conditioned on the observations $\mathbf{X}_m$. The edge connects an $\mathbf{X}_m$ node and a $Y_m$ node where $m = 1, \cdots, M$. The edge potential function is defined by:

$$\psi_e(Z, \mathcal{Y}, \mathcal{X}) = f_{yx}(Y_m, \mathbf{X}_m) = x_{m,Y_m} \tag{3}$$

where $x_{m,Y_m}$ is the $Y_m$-th component of the vector $\mathbf{X}_m$. A large weight for $(\mathbf{X}_m, Y_m)$ implies that the independent classifier trained on voxels of ROI $m$ is effective in giving correct predictions.

**Type-II Potential** $e = (Y_m, Y_l)$. Such edges model the dependencies between the ROIs. Note that not all pairs of ROIs are connected. The edge potential function is defined by:

$$\psi_e(Z, \mathcal{Y}, \mathcal{X}) = f_{yy}(Y_m, Y_l) = \begin{cases} \alpha, & Y_m = Y_l \\ 0, & Y_m \neq Y_l \end{cases} \tag{4}$$

where $\alpha > 0$. If two ROIs are connected, they tend to make similar predictions. A large weight for $(Y_m, Y_l)$ means the connection between $Y_m$ and $Y_l$ is strong.

**Type-III Potential** $e = (Z, Y_m)$. Such edges define a joint distribution over the class label and the prediction result of each ROI. The edge connects a $Y_m$ node and the $Z$ node where $m = 1, \cdots, M$.

The edge potential function is defined by:

$$\psi_e(Z, \mathcal{Y}, \mathcal{X}) = f_{yz}(Y_m, Z) = \begin{cases} \beta, & Y_m = Z \\ 0, & Y_m \neq Z \end{cases} \tag{5}$$

where $\beta > 0$. A large weight for $(Z, Y_m)$ means ROI $m$ has a big contribution to the top-level prediction of the brain.

Allowing connected ROIs to interact with each other makes our model significantly different from existing MVPA methods [15, 24, 18, 16], and can improve the prediction accuracy of each ROI. Intuitively, if the values of all components in $\mathbf{X}_m^i$ are similar, then ROI $m$ is likely to have incorrect predictions if its classifier merely relies on $\mathbf{X}_m^i$. In such situations it is possible for the classifier for one ROI to make better predictions if it can use the information in its connected ROIs.

## 2.2 Learning the Structural Connections of the Hidden Layer in HCRF Model

We have described a method that models the connections between ROIs to build a classification predictor on top of all ROIs. However, for many tasks (e.g. scene categorization), one critical scientific goal is to uncover which ROIs are functionally connected for that task. Automatic learning of the structures of graphical models is a difficult problem in machine learning. To illustrate the difficulty, let us assume that we have 4 ROIs and that we want to explore all possible models of connectivity between them. There are 6 possible connections between the ROIs, so in order to investigate whether all possible combinations of connections are present, we need to evaluate $2^6 = 64$ different models. For 5 ROIs we have 10 potential connections, leading to $2^{10} = 1024$. In general, given $M$ ROIs, there are $2^{M(M-1)/2}$ possible combinations of connections. In situations with many ROIs, evaluating all possible structures quickly becomes impractical because of the computational constraints. Approximate approaches to learn the structures of directed graphs use the generative process in the model [21, 19, 32]. For undirected graphs, it is usually assumed that the structures are pre-defined [29]. Some incremental approaches [26, 22] were proposed for random fields construction. However the computational complexity of these approaches is still high.

In our model shown in Fig.1, the potentials represented by solid lines are fixed (type-I and type-III). That is to say, each ROI always makes predictions based on the information in its voxels, and the response at the top level is always influenced by the prediction results of all ROIs. That leaves the dependencies between ROIs (type-II edges, the dashed line in Fig.1) to be learned. Therefore, our structural learning starts from a graphical model containing only type-I and type-III potentials, without any interactions between ROIs. Based on this initial model, we evaluate each type-II potential respectively to decide if it should be added to the model.

As we have described in Section 1, connections among ROIs play a key role in information processing. Executing a specific task (e.g., scene categorization) activates certain ROIs as well as rely on connections between some of them. Inspired by this fact, we evaluate whether two ROIs, say ROIs $m$ and $l$, should be connected by comparing two models with and without an edge between $Y_m$ and

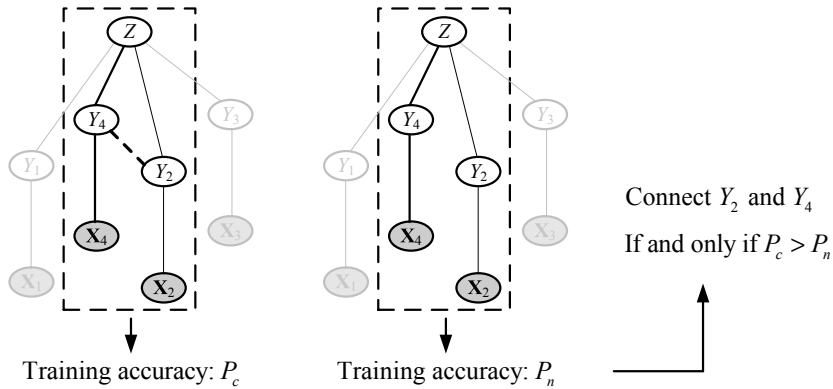

Figure 2: An illustration for evaluating if ROIs 2 and 4 should be connected. All other ROIs are omitted. We compare the performance of two modes with (left) and without (right) interactions between ROIs 2 and 4.

---

**Input**: $M$ ROIs and their feature vectors $\mathcal{X} = \{\mathbf{X}_1, \cdots, \mathbf{X}_M\}$. A HCRF model $\mathcal{G}$ with nodes $Z$,
   $Y_1, \cdots, Y_M, \mathbf{X}_1, \cdots, \mathbf{X}_M$, and edges $(Y_1, \mathbf{X}_1), \cdots, (Y_M, \mathbf{X}_M), (Z, Y_1), \cdots, (Z, Y_M)$.
**foreach** *pair of ROIs $m$ and $l$* **do**
 | Train an HCRF model with nodes $Z, Y_m, Y_l, \mathcal{X}_m, \mathcal{X}_l$, and edges $(Y_m, \mathbf{X}_m), (Y_l, \mathbf{X}_l), (Z, Y_m)$,
 |   $(Z, Y_l), (Y_m, Y_l)$.  Obtain training accuracy $P_c$;
 | Train an HCRF model with nodes $Z, Y_m, Y_l, \mathcal{X}_m, \mathcal{X}_l$, and edges $(Y_m, \mathbf{X}_m), (Y_l, \mathbf{X}_l), (Z, Y_m)$,
 |   $(Z, Y_l)$.  Obtain training accuracy $P_n$;
 | **if** $P_c > P_n$ **then** Add edge $(Y_m, Y_l)$ to the input model $\mathcal{G}$;
**Output**: The updated model $\mathcal{G}$.

---

**Algorithm 1**: The algorithm for uncovering structural connections between ROIs in the HCRF model.

$Y_l$. If allowing interactions between ROIs $m$ and $l$ helps to improve top-level recognition performance, thus more closely approximating human performance, then $m$ and $l$ should be connected. Furthermore, we ignore information in all other ROIs when evaluating the connection between ROIs $m$ and $l$ (Fig.2). So the model will only contain 5 nodes: $Z, Y_m, Y_l, \mathbf{X}_m$, and $\mathbf{X}_l$. Although some useful information might be lost compared to evaluating all possible combinations of connections, approximating the algorithm in this way can enable the evaluation of many possible connections in a reasonable amount of time, making this algorithm much more practical.

The structural learning algorithm is shown in Algorithm 1, and an illustration of evaluating the connection between ROI 2 and 4 is in Fig.2.

### 2.3 Model Learning and Inference

**Learning**   In the step of structural learning, we need to estimate model parameters to compare the models with or without a type-II connection (see Fig.2 for an illustration). Once we have determined which ROIs should interact, i.e. which type-II potentials should be set, we would like to find out the strength of these connections as well as type-I and III potentials. Here the parameters $\boldsymbol{\theta} = \{\theta_e\}_e$ are learned by maximizing the conditional log-likelihood of class label $Z$ on training data $\mathcal{X}$:

$$\boldsymbol{\theta}^* = \arg\max_{\boldsymbol{\theta}} L(\boldsymbol{\theta}) = \arg\max_{\boldsymbol{\theta}} \sum_i \log p(Z^i | \mathcal{X}^i; \boldsymbol{\theta})$$

$$= \arg\max_{\boldsymbol{\theta}} \sum_i \log \frac{\sum_{\mathcal{Y}} \exp(\Psi(Z^i, \mathcal{Y}, \mathcal{X}^i; \boldsymbol{\theta}))}{\sum_Z \sum_{\mathcal{Y}} \exp(\Psi(Z, \mathcal{Y}, \mathcal{X}^i; \boldsymbol{\theta}))} \tag{6}$$

The objective function is not concave due to the hidden variables $\mathcal{Y}$. Although finding the global optimum is difficult, we can still find a local optimum by iteratively updating the values of $\boldsymbol{\theta}$ using the gradient descent method. To be specific, we first set $\boldsymbol{\theta}$ to be initial values $\boldsymbol{\theta}^{(0)}$, and for each iteration we adopt the following formula to update $\boldsymbol{\theta}^{(n)}$ to $\boldsymbol{\theta}^{(n+1)}$:

$$\boldsymbol{\theta}^{(n+1)} = \boldsymbol{\theta}^{(n)} - \frac{\mathbf{G}(\boldsymbol{\theta}^{(n)})^\top \mathbf{G}(\boldsymbol{\theta}^{(n)})}{\mathbf{G}(\boldsymbol{\theta}^{(n)})^\top \mathbf{H}(\boldsymbol{\theta}^{(n)}) \mathbf{G}(\boldsymbol{\theta}^{(n)})} \mathbf{G}(\boldsymbol{\theta}^{(n)}) \tag{7}$$

where $\mathbf{G}(\boldsymbol{\theta})$ and $\mathbf{H}(\boldsymbol{\theta})$ are the gradient vector and Hessian matrix of $L(\boldsymbol{\theta})$ respectively. This iterative updating continues until reaching a maximum number of iterations or $\|\mathbf{G}(\boldsymbol{\theta})\|$ is smaller than a threshold. When the number of ROIs is large, marginalizing over all possible values of $\mathcal{Y}$ is time-consuming. In such situations we can use Gibbs sampling to compute the gradient vector and Hessian matrix of $L(\boldsymbol{\theta})$. In the case of natural scene categorization, evidence from neuroscience studies have postulated that 7 regions are likely to play critical roles in this task [31]. We therefore consider 7 ROIs in our experiment, allowing us to marginalize over all possible values of Y.

**Inference**   Given the model parameters $\boldsymbol{\theta}^*$ and a sample $\mathcal{X}$, the top-level prediction result is

$$Z^* = \arg\max_Z p(Z | \mathcal{X}; \boldsymbol{\theta}^*) \tag{8}$$

After $Z^*$ is obtained, we can get the prediction results corresponding to each ROI by

$$\mathcal{Y}^* = \arg\max_{\mathcal{Y}} p(Z^*, \mathcal{Y} | \mathcal{X}; \boldsymbol{\theta}^*) \tag{9}$$

# 3 Related Work

In this paper, we model the dependencies between ROIs in an HCRF framework, which improves the ROI-level as well as the top-level decoding accuracy by allowing ROIs to exchange information. Other approaches to inferring connections between brain regions from fMRI data can be broadly separated into effective connectivity and functional connectivity [11]. Models for effective connectivity, such as Granger causality mapping [14] and dynamic causal modeling [13], model directed connections between brain regions. These approaches were developed to account for biological temporal dependencies, which is not the case in this work. Functional connectivity refers to undirected connections, which can be either model-driven or data-driven [23]. Model-driven methods usually test a prior hypothesis by correlating the time courses of a seed voxel and a target voxel [12]. Data-driven methods, such as Independent Component Analysis [8], are typically used to identify spatial modes of coherent activity in the brain at rest.

None of these methods, however, has the ability to use the specific relation between the patterns of voxel activations inside ROIs and the ground truth of the experimental condition. The structural learning method proposed in this paper offers an entirely new way to assess the interactions between brain regions based on the exchange of information between ROIs so that the accuracy of decoding experimental conditions from the data is improved. Furthermore in contrast with the conventional model comparison approaches of trying to optimize the evidence of each model [2], our method relates the connectivity structure to observed brain activities as well as the classes of stimuli that elicited the activities. Therefore the model proposed here provides a novel and natural way to model the implicit dependencies between different ROIs.

# 4 Experimental Evaluation

## 4.1 Data Set and Experimental Design

In order to evaluate the proposed method we re-analyze the fMRI data set from our work in [31]. In this experiment, 5 subjects were presented with color images of 6 scene categories: beaches, buildings, forests, highways, industry, and mountains. Photographs were chosen to capture the high variability within each scene category. Images were presented in blocks of 10 images of the same category lasting for 16 seconds (8 brain acquisitions). Each subject performed 12 runs, with each run containing one block for each of the six categories. Please refer to [31] for more details.

We use 7 ROIs that are likely to play critical roles for natural scene categorization. They were determined in separate localizer scans: V1, left/right LOC, left/right PPA, left/right RSC. The data for two subjects were excluded, because not all of the ROIs could be found in the localizer scans for these subjects. For the analysis we use two nested cross validations over the 12 runs for each subject. In the outer loop we cross-validate on each subject to test the performance of the proposed method. For each subject, 11 runs out of 12 are selected as training samples and the remaining run is used as the testing set. For each subject this procedure is repeated 12 times, in turn leaving each run out for testing once. Average accuracy of the 36 experiments across all subjects is used to evaluate the performance of the model. In the inner loop, we use 10 of the 11 training runs to train an SVM classifier for each ROI and each subject, and the remaining run to learn the connections between ROIs and train the HCRF model by using outputs of the SVM classifiers. We repeat this procedure 11 times, giving us 11 models. Results of the 11 models on the test data in the inner loop are combined using bagging [4]. We empirically set both $\alpha$ in Equ.(4) and $\beta$ in Equ.(5) to 0.5.

## 4.2 Scene Classification Results and Analysis

In order to comprehensively evaluate the performance of the proposed structural learning and modeling approach, we consider different settings of the intermediate layer of our HCRF model. While always keeping all type-I and type-III potentials connected, we consider five different dependencies between the ROIs as shown in Fig.3. The setting in Fig.3(e) possesses all properties of our method: the connections between ROIs are determined by structural learning, and the weights of the connections are obtained by estimating model parameters in Equ.(6). In order to estimate the effectiveness of our structural learning method, we compare this setting with the situations where no connections exists between any of the ROIs (Fig.3(a)), and all ROIs are fully connected (Fig.3(b,c)). In each connectivity situation, we either use the same (Fig.3(b,d)) or different (Fig.3(c,e)) weights for type-II

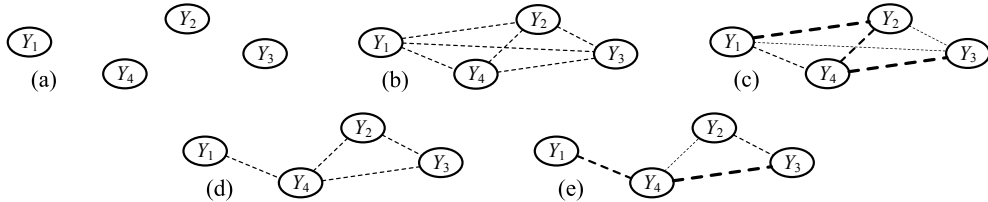

Figure 3: Various settings of the intermediate layer of our model. Dashed lines represent type-II potentials. In each setting we keep all type-I and III potentials connected. For simplicity, we omit the visualizations of type-I and III potentials here. Different line widths represent different potential weights. **(a)** No connection exists between any pair of ROIs. **(b,c)** The ROIs are fully connected. **(d,e)** The connections between ROIs are obtained by structural learning. **(b,d)** All type-II potentials have equal weights. **(c,e)** The weights of different type-II potentials can be different. Note that **(e)** is the full model in this paper.

Table 1: Recognition accuracy for predicting natural scene categories with different methods (chance is 1/6). "Overall classification" means the accuracy for predicting the categories by the top-level node in Fig.1. We carry out experiments on the HCRF models with different settings of the type-II potentials, as shown in Fig.3. Note that we always learn the weights of type-I and type-III potentials. We also list classification results of the SVM classifiers independently trained on each ROI as the baseline. The bolded numbers indicate superior performance compared to all other settings for each ROI. $^*p < 0.01$; $^{**}p < 0.005$.

| | Method | SVM | Fig.3(a) | Fig.3(b) | Fig.3(c) | Fig.3(d) | Fig.3(e) |
|---|---|---|---|---|---|---|---|
| | **Overall classification** | N/A | 31%$^*$ | 29%$^*$ | 33%$^{**}$ | 34%$^{**}$ | **36%**$^{**}$ |
| | V1 | 21% | 22% | 25% | 24% | 27% | **28%**$^*$ |
| | left LOC | 22% | 23% | 27% | 29%$^*$ | 31%$^*$ | **32%**$^{**}$ |
| | right LOC | 25% | 24% | 27% | 30%$^*$ | 29%$^*$ | **33%**$^{**}$ |
| **ROI** | left PPA | 27% | 27% | 26% | 28%$^*$ | **31%**$^*$ | **31%**$^*$ |
| | right PPA | 26% | 28%$^*$ | 28%$^*$ | 31%$^*$ | 31%$^*$ | **32%**$^{**}$ |
| | left RSC | 30% | 30%$^*$ | 30%$^*$ | 32%$^*$ | 33%$^{**}$ | **35%**$^{**}$ |
| | right RSC | 26%$^*$ | 27% | 29%$^*$ | 30%$^*$ | 30%$^*$ | **32%**$^{**}$ |

potentials. Note that the type-II potentials of the models in Fig.3(b,d) are also obtaining by learning. Classification accuracy of the five different HCRF models, along with individual SVM classification accuracy for each ROI, is shown in Tbl.1. Note that the model with no type-II potentials (Fig.3(a)) is different from independent SVM classifiers because of the type-I potentials.

From Table 1 it becomes clear that learning both the structure of the connections and their strengths leads to more improvement in decoding accuracy than either one of these alone. The overall, top-level classification rate increases from 31% for the variant of the model without any connections (Fig.3(a)) to 36% for the variant with the structure of the model as well as the connection strengths learned (Fig.3(e)). We see similar improvements for the individual ROIs: 4-5% for PPA and RSC, 6% for V1, and 9% for LOC. The fact that decoding from LOC benefits most from interacting with other ROIs is interesting and significant. We will discuss this finding in more detail below.

### 4.3 Structural Learning Results and Analysis

Having established that our full HCRF model outperforms other comparison models in the recognition task, we now investigate how our model can shed light on learning connectivity between brain regions. In the nested cross validation procedure, 12×11=132 structural maps are learned for each subject. Tbl.2 reports for each subject which connections are present in what fraction of these structural maps. A connection is regarded as a strong connection for a subject if it presents in at least half of the models learned for this subject. In Tbl.2 we use larger font size to denote the connections which are strong on more subjects. Connections that are strong for all subjects are marked in bold.

We see that both LOC and PPA show strong interactions between the contralateral counterparts, which makes sense for integrating information across the visual hemifields. We also observe strong interactions between PPA and RSC across hemispheres, which underscores the importance of across-hemifield integration of visual information. We see a similar effect in the interactions between LOC and PPA: strong contralateral interactions. Left LOC also interacts strongly with right RSC.

Table 2: Statistics of structural connections. For each subject we have 132 learned structural maps (12-fold cross-validation, each one has 11 models). This table shows the percentage of the times that the corresponding connection is learned in the 132 experiments. Larger font size denotes connections that are strong on more subjects. Connections that are strong on all subjects are marked in bold.

| Connection | Sbj.1 | Sbj.2 | Sbj.3 | Connection | Sbj.1 | Sbj.2 | Sbj.3 |
|---|---|---|---|---|---|---|---|
| V1-leftLOC | 0.67 | 0.25 | 0.33 | **rightLOC-leftPPA** | **0.58** | **0.58** | **0.66** |
| V1-rightLOC | 0.50 | 0.29 | 0.54 | rightLOC-rightPPA | 0.36 | 0.58 | 0.89 |
| V1-leftPPA | 0.44 | 0.29 | 0.36 | rightLOC-leftRSC | 0.63 | 0.38 | 0.31 |
| V1-rightPPA | 0.38 | 0.33 | 0.69 | rightLOC-rightRSC | 0.36 | 0.30 | 0.87 |
| V1-leftRSC | 0.29 | 0.30 | 0.23 | **leftPPA-rightPPA** | **0.99** | **0.56** | **0.78** |
| V1-rightRSC | 0.36 | 0.29 | 0.59 | leftPPA-leftRSC | 0.97 | 0.34 | 0.46 |
| **leftLOC-rightLOC** | **0.66** | **0.88** | **0.71** | leftPPA-rightRSC | 0.61 | 0.53 | 0.40 |
| leftLOC-leftPPA | 0.46 | 0.64 | 0.76 | **rightPPA-leftRSC** | **0.67** | **0.74** | **0.51** |
| **leftLOC-rightPPA** | **0.75** | **0.96** | **0.65** | rightPPA-rightRSC | 0.93 | 0.74 | 0.41 |
| leftLOC-leftRSC | 0.41 | 0.78 | 0.61 | leftRSC-rightRSC | 0.65 | 0.20 | 0.45 |
| **leftLOC-rightRSC** | **0.75** | **0.83** | **0.76** | | | | |

The strong interactions between PPA and RSC are not surprising, since both are typically associated with the processing of natural scenes [25], albeit with slightly different roles [7]. The interactions between LOC and PPA are somewhat more surprising, since LOC is usually associated with the processing of isolated objects. Together with the strong improvement of decoding accuracy for natural scene categories from LOC when it is allowed to interact with other ROIs (see above), this suggests a role for LOC in scene categorization. It is conceivable that the detection of typical objects (e.g., a car) helps with determining the scene category (e.g., highway), as has been shown in [17, 6]. On the other hand, it is also possible that information flows the other way, that scene-specific information in PPA and RSC feeds into LOC to bias object detection based on the scene category (see [3, 1]), and that the classifier decodes this bias signal in LOC. Fig.4 shows the connections which are strong on at least two subjects.

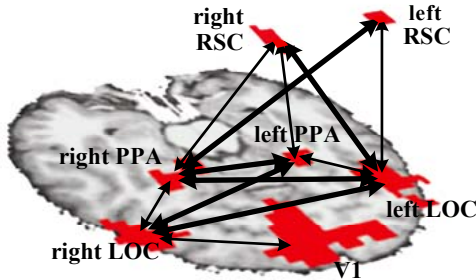

Figure 4: Schematic illustration of the connections between the seven ROIs obtained by our structural learning method. Activated regions for the seven ROIs are marked in red. The connections shown in this figure are strong on at least two of the three subjects. Connections that are strong for all three subjects (marked with bold in Table 2) are marked with thicker lines in this figure.

## 5 Conclusion

In this paper we modeled the interactions between brain regions in an HCRF framework. We also presented a structural learning method to automatically uncover the connections between ROIs. Experimental results showed that our approach can improve the top-level as well as ROI-level prediction accuracy, as well as uncover some meaningful connections between ROIs. One direction for future work is to use an exploratory "searchlight" approach [20] to automatically discover ROIs, and apply our structural learning and modeling method to those ROIs.

**Acknowledgements**

This work is funded by National Institutes of Health Grant 1 R01 EY019429 (to L.F.-F., D.M.B., D.B.W.), a Beckman Postdoctoral Fellowship (to D.B.W.), a Microsoft Research New Faculty Fellowship (to L.F.-F.), and the Frank Moss Gift Fund (to L.F-F.). The authors would like to thank Barry Chai, Linjie Luo, and Hao Su for helpful comments and discussions.

# References

[1] M. Bar. Visual objects in context. *Nature Rev Neurosci*, 5(8):617–629, 2004.

[2] D. Barber and C. M. Bishop. Bayesian model comparison by monte carlo chaining. In *NIPS*, 1997.

[3] I. Biederman. Perceiving real-world scenes. *Science*, 177(4043):77–80, 1972.

[4] L. Breiman. Bagging predictors. *Mach Learn*, 24:123–140, 1996.

[5] B. Chai†, D. B. Walther†, D. M. Beck*, and L. Fei-Fei*. Exploring functional connectivities of the human brain using multivariate information analysis. In *NIPS*, 2009. (†,* indicates equal contribution).

[6] J. L. Davenport and M. C. Potter. Scene consistency in object and background perception. *Psychol Sci*, 15(8):559–564, 2004.

[7] R. A. Epstein and J. S. Higgins. Differential parahippocampal and retrosplenial involvement in three types of scene recognition. *Cereb Cortex*, 17:1680–1693, 2007.

[8] F. Esposito, E. Formisano, E. Seifritz, R. Geobel, R. Morrone, G. Tedeschi, and F. D. Salle. Spatial independent component analysis of functional MRI time-series: To what extent do results depend on the algorithm used. *Hum Brain Mapp*, 16:146–157, 2002.

[9] L. Fei-Fei, A. Iyer, C. Koch, and P. Perona. What do we perceive in a glance of a real-world scene? *J Vision*, 7(1):1–29, 2007.

[10] D. J. Felleman and D. C. van Essen. Distributed hierarchical processing in the primate cerebral cortex. *Cereb Cortex*, 1:1–47, 1991.

[11] K. J. Friston. Functional and effective connectivity in neuroimaging: a synthesis. *Hum Brain Mapp*, 2:56–78, 1995.

[12] K. J. Friston, C. Frith, F. P. Liddle, and R. Frackowiak. Functional connectivity: The principal-component analysis of large (PET) data sets. *J Cerebr Blood F Met*, 13:5–14, 1993.

[13] K. J. Friston, L. Harrison, and W. Penny. Dynamic causal modeling. *NeuroImage*, 19:1273–1302, 2003.

[14] R. Goebel, A. Roebroeck, D.-S. Kim, and E. Formisano. Investigating directed cortical interactions in time-resolved fMRI data using vector autoregressive modeling and granger causality mapping. *Magn Reson Imaging*, 21:1251–1261, 2003.

[15] J. V. Haxby, M. I. Gobbini, M. L. Furey, A. Ishai, J. Schouten, and P. Pietrini. Distributed and overlapping representations of faces and objects in ventral temporal cortex. *Science*, 293(5539):2425–2430, 2001.

[16] J.-D. Haynes and G. Rees. Predicting the orientation of invisible stimuli from activity in human primary visual cortex. *Nat Neurosci*, 8:686–691, 2005.

[17] A. Hollingworth and J. M. Henderson. Accurate visual memory for previously attended objects in natural scenes. *J Exp Psychol Human*, 28:113–136, 2002.

[18] Y. Kamitani and F. Tong. Decoding the visual and subjective contents of the human brain. *Nat Neurosci*, 8:679–685, 2005.

[19] C. Kemp and J. B. Tenenbaum. The discovery of structural form. *P Natl Acad Sci USA*, 105(31):10687–10692, 2008.

[20] N. Kriegeskorte, R. Goebel, and P. Bandettini. Information-based functional brain mapping. *P Natl Acad Sci USA*, 103(10):3863–3868, 2006.

[21] W. Lam and F. Bacchus. Learning Bayesian belief networks: An approach based on the mdl principle. *Comput Intell*, 10(4):269–293, 1994.

[22] S. Lee, V. Ganapahthi, and D. Koller. Efficient structure learning of markov networks using $l_1$-regularization. In *NIPS*, 2006.

[23] K. Li, L. Guo, J. Nie, G. Li, and T. Liu. Review of methods for functional brain connectivity detection using fmri. *Comput Med Imag Grap*, 33:131–139, 2009.

[24] D. Neill, A. Moore, F. Pereira, and T. Mitchell. Detecting significant multidimensional spatial clusters. In *NIPS*, 2004.

[25] K. O'Craven and N. Kanwisher. Mental imagery of faces and places activates corresponding stimulus-specific brain regions. *J Cognitive Neurosci*, 12:1013–1023, 2000.

[26] S. D. Pietra, V. D. Pietra, and J. Lafferty. Inducing features of random fields. *IEEE T Pattern Anal*, 19(4):380–393, 1997.

[27] M. C. Potter. Short-term conceptual memory for pictures. *J Exp Psychol - Hum L*, 2(5):509–522, 1976.

[28] A. Quattoni, S. Wang, L.-P. Morency, M. Collins, and T. Darrell. Hidden conditional random fields. *IEEE T Pattern Anal*, 29(10):1848–1852, 2007.

[29] B. Taskar, P. Abbeel, and D. Koller. Discriminative probabilistic models for relational data. In *UAI*, 2002.

[30] B. Tversky and K. Hemenway. Categories of scenes. *Cognitive Psychol*, 15:121–149, 1983.

[31] D. B. Walther, E. Caddigan, L. Fei-Fei*, and D. M. Beck*. Natural scene categories revealed in distributed patterns of activity in the human brain. *J Neurosci*, 29(34):10573–10581, 2009. (* indicates equal contribution).

[32] M. L. Wong, W. Lam, and K. S. Leung. Using evolutionary programming and minimum description length principle for data mining of bayesian networks. *IEEE T Pattern Anal*, 21(2):174–178, 1999.

